# Centric Models of the Orientation Map in Primary Visual Cortex

William Baxter
Department of Computer Science, S.U.N.Y. at Buffalo, NY 14620

Bruce Dow
Department of Physiology, S.U.N.Y. at Buffalo, NY 14620

## Abstract

In the visual cortex of the monkey the horizontal organization of the preferred orientations of orientation-selective cells follows two opposing rules: 1) neighbors tend to have similar orientation preferences, and 2) many different orientations are observed in a local region. Several orientation models which satisfy these constraints are found to differ in the spacing and the topological index of their singularities. Using the rate of orientation change as a measure, the models are compared to published experimental results.

## Introduction

It has been known for some years that there exist orientation-sensitive neurons in the visual cortex of cats and monkeys[1,2]. These cells react to highly specific patterns of light occurring in narrowly circumscribed regions of the visual field, i.e., the cell's receptive field. The best patterns for such cells are typically not diffuse levels of illumination, but elongated bars or edges oriented at specific angles. An individual cell responds maximally to a bar at a particular orientation, called the preferred orientation. Its response declines as the bar or edge is rotated away from this preferred orientation.

Orientation-sensitive cells have a highly regular organization in primary cortex[3]. Vertically, as an electrode proceeds into the depth of the cortex, the column of tissue contains cells that tend to have the same preferred orientation, at least in the upper layers. Horizontally, as an electrode progresses across the cortical surface, the preferred orientations change in a smooth, regular manner, so that the recorded orientations appear to rotate with distance. It is this horizontal structure we are concerned with, hereafter referred to as the orientation map. An orientation map is defined as a two-dimensional surface in which every point has associated with it a preferred orientation ranging from $0° \cdots 180°$. In discrete versions, such as the array of cells in the cortex or the discrete simulations in this paper, the orientation map will be considered to be a sampled version of the underlying continuous surface. The investigations of this paper are confined to the upper layers of macaque striate cortex.

Detailed knowledge of the two-dimensional layout of the orientation map has implications for the architecture, development, and function of the visual cortex. The organization of orientation-sensitive cells reflects, to some degree, the organization of intracortical connections in striate cortex. Plausible orientation maps can be generated by models with lateral connections that are uniformly exhibited by all cells in the layer[4,5], or by models which presume no specific intracortical connections, only appropriate patterns of afferent input[6]. In this paper, we examine models in which intracortical connections produce the orientation map but the orientation-controlling circuitry is not displayed by all cells. Rather, it derives from localized "centers" which are distributed across the cortical surface with uniform spacing[7,8,9].

The orientation map also represents a deformation in the retinotopy of primary visual cortex. Since the early sixties it has been known that V1 reflects a topographic map of the retina and hence the visual field[10]. There is some global distortion of this mapping[11,12,13], but generally spatial relations between points in the visual field are maintained on the cortical surface. This well-known phenomenon is only accurate for a medium-grain description of V1, however. At a finer cellular level there is considerable scattering of receptive fields at a given cortical location[14]. The notion of the hypercolumn[3] proposes that such scattering permits each region of the visual field to be analyzed by a population of cells consisting of all the necessary orientations and with inputs from both eyes. A quantitative description of the orientation map will allow prediction of the distances between iso-orientation zones of a particular orientation, and suggest how much cortical machinery is being brought to bear on the analysis of a given feature at a given location in the visual field.

## Models of the Orientation Map

### Hubel and Wiesel's Parallel Stripe Model

The classic model of the orientation map is the parallel stripe model first published by Hubel and Wiesel in 1972[15]. This model has been reproduced several times in their publications[3,16,17] and appears in many textbooks. The model consists of a series of parallel slabs, one slab for each orientation, which are postulated to be orthogonal to the ocular dominance stripes. The model predicts that a microelectrode advancing tangentially (i.e., horizontally) through the tissue should encounter steadily changing orientations. The rate of change, which is also called the orientation drift rate[18], is determined by the angle of the electrode with respect to the array of orientation stripes.

The parallel stripe model does not account for several phenomena reported in long tangential penetrations through striate cortex in macaque monkeys[17,19]. First, as pointed out by Swindale[20], the model predicts that some penetrations will have flat or very low orientation drift rates over lateral distances of hundreds of micrometers. This is because an electrode advancing horizontally and perpendicular to the ocular dominance stripes (and therefore parallel to the orientation stripes) would be expected to remain within a single orientation column over a considerable distance with its orientation drift rate equal to zero. Such results have never been observed. Second, reversals in the direction of the orientation drift, from clockwise to counterclockwise or vice versa, are commonly seen, yet this phenomenon is not addressed by the parallel stripe model. Wavy stripes in the ocular dominace system[21] do not by themselves introduce reversals. Third, there should be a negative correlation between the orientation drift rate and the ocularity "drift rate". That is, when orientation is changing rapidly, the electrode should be confined to a single ocular dominance stripe (low ocularity drift rate), whereas when ocularity is changing rapidly the electrode should be confined to a single orientation stripe (low orientation drift rate). This is clearly not evident in the recent studies of Livingstone and Hubel[17] (see especially their figs. 3b, 21 & 23), where both orientation and ocularity often have high drift rates in the same electrode track, i.e., they show a positive correlation. Anatomical studies with 2-deoxyglucose also fail to show that the orientation and ocular dominance column systems are orthogonal[22].

## Centric Models and the Topological Index

Another model, proposed by Braitenberg and Braitenberg in 1979[7], has the orientations arrayed radially around centers like spokes in a wheel The centers are spaced at distances of about 0.5mm. This model produces reversals and also the sinusoidal progressions frequently encountered in horizontal penetrations. However this approach suggests other possibilities, in fact an entire class of centric models. The organizing centers form discontinuities in the otherwise smooth field of orientations. Different topological types of discontinuity are possible, characterized by their topological index[23]. The topological index is a parameter computed by taking a path around a discontinuity and recording the rotation of the field elements (figure 1). The value of the index indicates the amount of rotation; the sign indicates the direction of rotation. An index of 1 signifies that the orientations rotate through 360°; an index of ½ signifies a 180° rotation. A positive index indicates that the orientations rotate in the same sense as a path taken around the singularity; a negative index indicates the reverse rotation.

Topological singularities are stable under orthogonal transformations, so that if the field elements are each rotated 90° the index of the singularity remains unchanged. Thus a +1 singularity may have orientations radiating out from it like spokes from a wheel, or it may be at the center of a series of concentric circles. Only four types of discontinuities are considered here, +1, -1, +½, −½, since these are the most stable, i.e., their neighborhoods are characterized by smooth change.

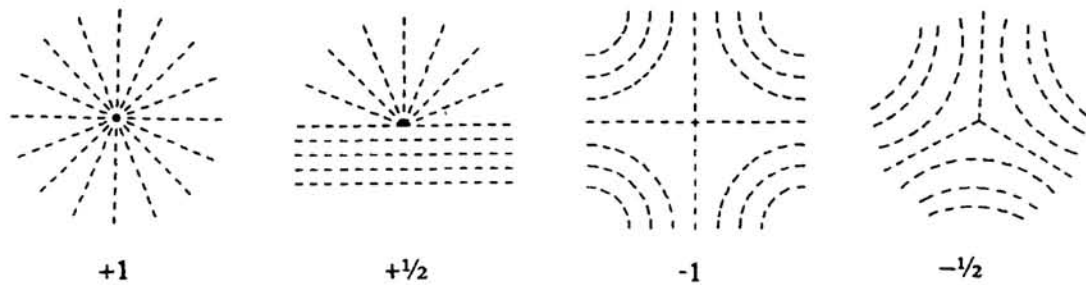

| +1 | +½ | -1 | −½ |

figure 1. Topological singularities. A positive index indicates that the orientations rotate in the same direction as a path taken around the singularity; a negative index indicates the reverse rotation. Orientations rotate through 360° around ±1 centers, 180° around ±½ centers.

## Cytochrome Oxidase Puffs

At topological singularities the change in orientation is discontinuous, which violates the structure of a smoothly changing orientation map; modellers try to minimize discontinuities in their models in order to satisfy the smoothness constraint. Interestingly, in the upper layers of striate cortex of monkeys, zones with little or no orientation selectivity have been discovered. These zones are notable for their high cytochrome oxidase reactivity[24] and have been referred to as cytochrome oxidase puffs, dots, spots, patches or blobs[17,25,26,27]. We will refer to them as puffs. If the organizing centers of centric models are located in the cytochrome oxidase puffs then the discontinuities in the orientation map are effectively eliminated (but see below). Braitenberg has indicated[28] that the +1 centers of his model should correspond to the puffs. Dow and Bauer proposed a model[8] with +1 and -1 centers in alternating puffs. Gotz proposed a similar model[9] with alternating +½ and −½ centers in the puffs. The last two models manage to eliminate all discontinuities from the interpuff zones, but they

assume a perfect rectangular lattice of cytochrome oxidase puffs.

## A Set of Centric Models

There are two parameters for the models considered here. (1) Whether the positive singularities are placed in every puff or in alternate puffs; and (2) whether the singularities are ±1's or ±½'s. This gives four centric models (figure 2):

E1 : +1 centers in puffs, -1 centers in the interpuff zones.
A1 : both +1 and -1 centers in the puffs, interdigitated in a checkerboard fashion.
E½ : +½ centers in the puffs, −½ centers in the interpuff zones.
A½ : both +½ and −½ centers in the puffs, as in A1.

The E1 model corresponds to the Braitenberg model transposed to a rectangular array rather than an hexagonal one, in accordance with the observed organization of the cytochrome oxidase regions[27]. In fact, the rectangular version of the Braitenberg model is pictured in figure 49 of[27]. The A1 model was originally proposed by Dow and Bauer[8] and is also pictured in an article by Mitchison[29]. The A½ model was proposed by Gotz[9]. It should be noted that the E1 and A1 models are the same model rotated and scaled a bit; the E½ and A½ have the same relationship.

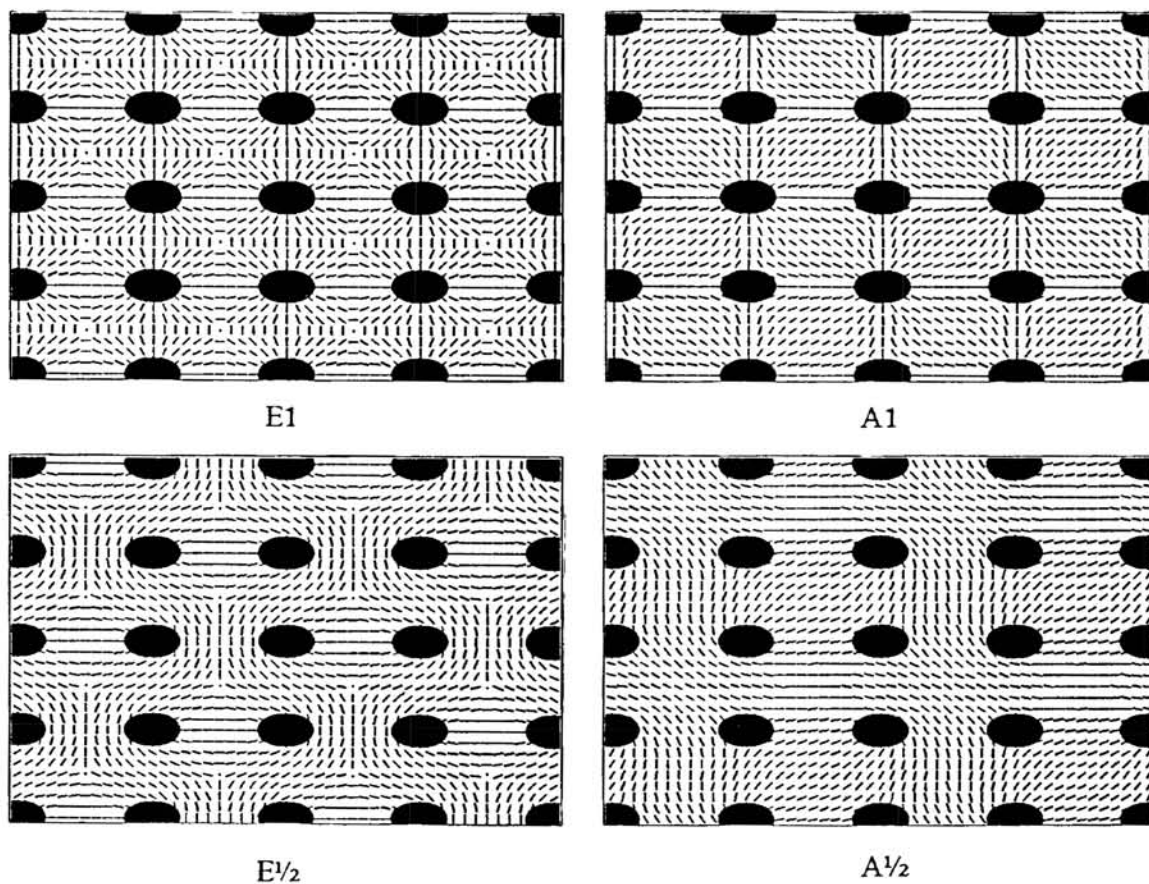

E1

A1

E½

A½

figure 2. The four centric models. Dark ellipses represent cytochrome oxidase puffs. Dots in interpuff zones of E1 & E½ indicate singularities at those points.

*Simulations*

Simulated horizontal electrode recordings were made in the four models to compare their orientation drift rates with those of published recordings. In the computer simulations (figure 2) the interpuff distances were chosen to correspond to histological measurements[27]. Puff centers are separated by $500\mu$ along their long axes, $350\mu$ along the short axes. The density of the arrays was chosen to approximate the sampling frequency observed in Hubel and Wiesel's horizontal electrode recording experiments[19], about 20 cells per millimeter. Therefore the cell density of the simulation arrays was about six times that shown in the figure.

All of the models produce simulated electrode data that qualitatively resemble the published recording results, e.g., they contain reversals, and runs of constantly changing orientations. The orientation drift rate and number of reversals vary in the different models.

The models of figure 2 are shown in perfectly rectangular arrays. Some important characteristics of the models, such as the absence of discontinuites in interpuff zones, are dependent on this regularity. However, the real arrangement of cytochrome oxidase puffs is somewhat irregular, as in Horton's figure 3[27]. A small set of puffs from the parafoveal region of Horton's figure was enlarged and each of the centric models was embedded in this irregular array. The E1 model and a typical simulated electrode track are shown in figure 3. Several problems are encountered when models developed in a regular lattice are implemented in the irregular lattice of the real system; the models have appreciably different properties. The -1 singularities in E1's interpuff zones have been reduced to $-\frac{1}{2}$'s; the A1 and A$\frac{1}{2}$ models now have some interpuff discontinuities where before they had none.

## Quantitative Comparisons

*Measurement of the Orientation Drift Rate*

There are two sets of centric models in the computer simulations: a set in the perfectly rectangular array (figure 2) and a set in the irregular puff array (as in figure 3). At this point we can generate as many tracks in the simulation arrays as we wish. How can this information be compared to the published records? The orientation drift rate, or slope, is one basis for distinguishing between models. In real electrode tracks however, the data are rather noisy, perhaps from the measuring process or from inherent unevenness of the orientation map. The typical approach is to fit a straight line and use the slope of this line. Reversals in the tracks require that lines be fit piecewise, the approach used by Hubel and Wiesel[19]. Because of the unevenness of the data it is not always clear what constitutes a reversal. Livingstone and Hubel[17] report that the track in their figure 5 has only two reversals in 5 millimeters. Yet there seem to be numerous microreversals between the 1st and 3rd millimeter of their track. At what point is a change in slope considered a true reversal rather than just noise?

The approach used here was to use a local slope measure and ignore the problem of reversals - this permitted the fast calculation of slope by computer. A single electrode track, usually several millimeters long, was assigned a single slope, the average of the derivative taken at each point of the track. Since these are discrete samples, the local derivative must be approximated by taking measurements over a small neighborhood. How large should this neighborhood be? If too small it will be susceptible to noise in the orientation measures, if too large it will "flatten out" true reversals. Slope

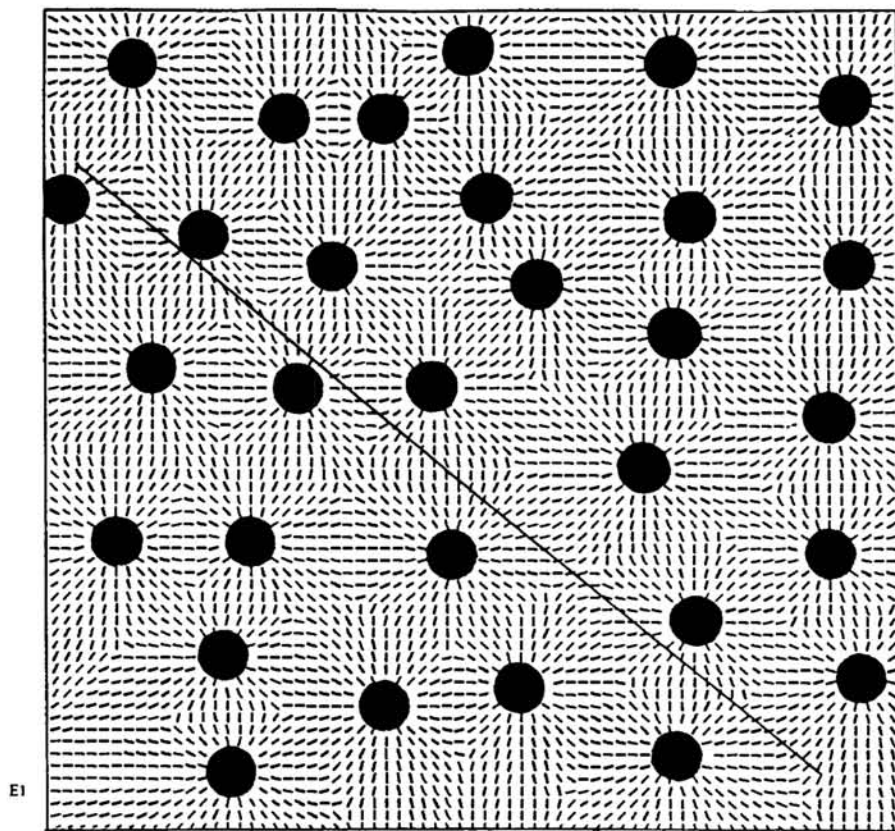

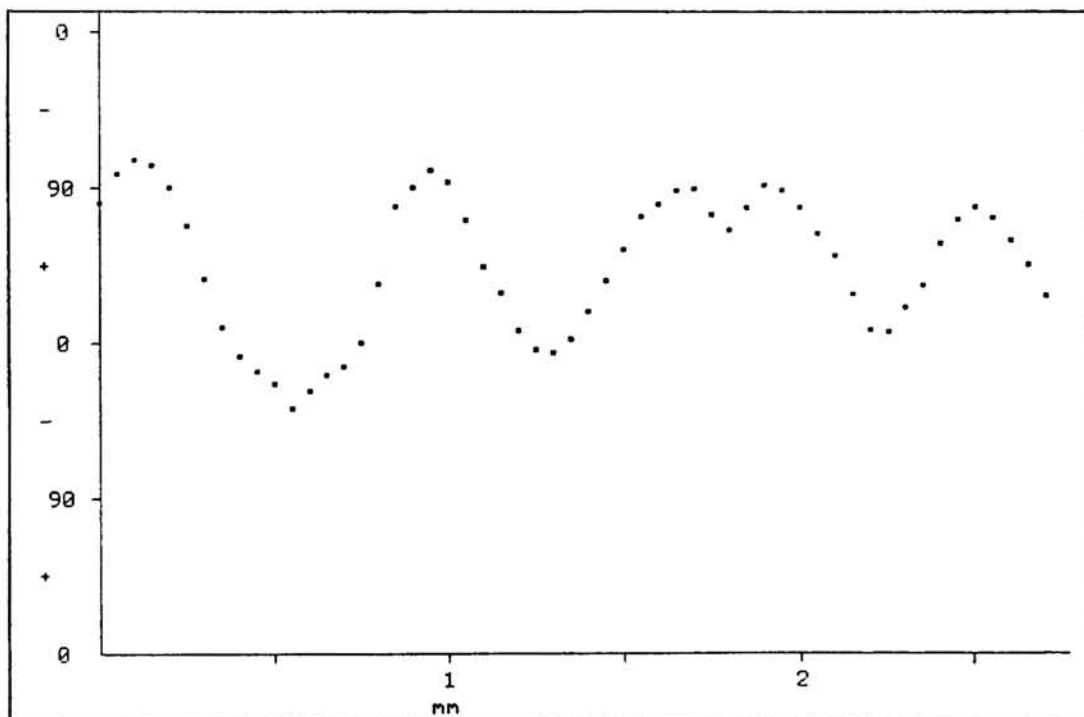

figure 3. A centric model in a realistic puff array (from[27]). A simulated electrode track and resulting data are shown. Only the E1 model is shown here, but other models were similarly embedded in this array.

measures using neighborhoods of several sizes were applied to six published horizontal electrode tracks from the foveal and parafoveal upper layers of macaque striate cortex: figures 5,6,7 from[17], figure 16 from[3], figure 1 from[30]. A neighborhood of 0.1mm, which attempts to fit a line between virtually every pair of points, gave abnormally high slopes. Larger neighborhoods tended to give lower slopes, especially to those tracks which contained reversals. The smallest window that gave consistent measures for all six tracks was 0.2mm; therefore this window was chosen for comparisons between published data and the centric models. This measure gave an average slope of 285 degrees per millimeter in the six published samples of track data, compared to Hubel & Wiesel's measure of 281 deg/mm for the penetrations in their 1974 paper[19].

*Slope measures of the centric models*

The slope measure was applied to several thousand tracks at random locations and angles in the simulation arrays, and a slope was computed for each simulated electrode track. Average slopes of the models are shown in Table I. Generally, models with ±1 centers have higher slopes than those with ±½ centers; models with centers in every puff have higher slopes than the alternate puff models. Thus E1 showed the highest orientation drift rate, A½ the lowest, with A1 and E½ having intermediate rates. The E1 model, in both the rectangular and irregular arrays, produced the most realistic slope values.

TABLE I   Average slopes of the centric models

|      | Rectangular array | Irregular array |
|------|-------------------|-----------------|
| E1   | 312               | 289             |
| A1   | 216               | 216             |
| E½   | 198               | 202             |
| A½   | 117               | 144             |

Numbers are in degrees/mm. Slope measure (window = 0.2mm) applied
to 6 published records yielded an average slope of 285 degrees/mm.

Discussion

*Constraints on the Orientation Map*

Our original definition of the orientation map permits each cell to have an orientation preference whose angle is completely independent of its neighbors. But this is much too general. Looking at the results of tangential electrode penetrations, there are two striking constraints in the data. The first of these is reflected in the smoothness of the graphs. Orientation changes in a regular manner as the electrode moves horizontally through the upper layers: neighboring cells have similar orientation preferences. Discontinuities do occur but are rare. The other constraint is the fact that the orientation is always changing with distance, although the rate of change may vary. Sequences of constant orientation are very rare and when they do occur they never carry on for any appreciable distance. This is one of the major reasons why the parallel stripe model is untenable. The two major constraints on the orientation map may be put informally as follows:

1. The smoothness constraint : neighboring points have similar orientation preferences.

2. The heterogeneity constraint : all orientations should be represented within a small region of the cortical surface.

This second constraint is a bit stronger than the data imply. The experimental results only show that the orientations change regularly with distance, not that all orientations must be present within a region. But this constraint is important with respect to visual processing and the notion of hypercolumns[3].

These are opposing constraints: the first tends to minimize the slope, or orientation drift rate, while the second tends to maximize this rate. Thus the organization of the orientation map is analogous to physical systems that exhibit "frustration", that is, the elements must satisfy conflicting constraints[31]. One of the properties of such systems is that there are many near-optimal solutions, no one of which is significantly better than the others. As a result, there are many plausible orientation maps: any map that satisfies these two constraints will generate qualitatively plausible simulated electrode tracks. This points out the need for quantitative comparisons between models and experimental results.

*Centric models and the two constraints*

What are some possible mechanisms of the constraints that generate the orientation map? Smoothness is a local property and could be attributed to the workings of individual cells. It seems to be a fundamental property of cortex that adjacent cells respond to similar stimuli. The heterogeneity requirement operates at a slightly larger scale, that of a hypercolumn rather than a minicolumn. While the first constraint may be modeled as a property of individual cells, the second constraint is distributed over a region of cells. How can such a collection of cells insure that its members cycle through all the required orientations? The topological singularities discussed earlier, by definition, include all orientations within a restricted region. By distributing these centers across the surface of the cortex, the heterogeneity constraint may be satisfied. In fact, the amount of orientation drift rate is a function of the density of this distribution (i.e., more centers per unit area give higher drift rates).

It has been noted that the E1 and the A1 organizations are the same topological model, but on different scales; the low drift rates of the A1 model may be increased by increasing the density of the +1 centers to that of the E1 model. The same relationship holds for the E½ and A½ models. It is also possible to obtain realistic orientation drift rates by increasing the density of +½ centers, or by mixing +1's and +½'s. However, these alternatives increase the number of interpuff singularities. And given the possible combinations of centers, it may be more than coincidental that a set of +1 centers at just the spacing of the cytochrome oxidase regions results in realistic orientation drift rates.

*Cortical Architecture and Types of Circuitry*

Thus far, we have not addressed the issue of how the preferred orientations are generated. The mechanism is presently unknown, but attempts to depict it have traditionally been of a geometric nature, alluding to the dendritic morphology[1,8,28,32]. More recently, computer simulations have shown that orientation-sensitive units may be obtained from asymmetries in the receptive fields of afferents[6], or developed using

simple Hebbian rules for altering synaptic weights[5]. That is, given appropriate network parameters, orientation tuning arises an as inherent property of some neural networks. Centric models propose a quite different approach in which an originally untuned cell is "programmed" by a center located at some distance to respond to a specific orientation. So, for an individual cell, does orientation develop locally, or is it "imposed from without"? Both of these mechanisms may be in effect, acting synergistically to produce the final orientation map. The map may spontaneously form on the embryonic cortex, but with cells that are nonspecific and broadly tuned. The organization imposed by the centers could have two effects on this incipient map. First, the additional influence from centers could "tighten up" the tuning curves, making the cells more specific. Second, the spacing of the centers specifies a distinct and uniform scale for the heterogeneity of the map. An unsupervised developing orientation map could have broad expanses of iso-orientation zones mixed with regions of rapidly changing orientations. The spacing of the puffs, hence the architecture of the cortex, insures that there is an appropriate variety of feature sensitive cells at each location. This has implications for cortical functioning: given the distances of lateral connectivity, for a cell of a given orientation, we can estimate how many other iso-orientation zones of that same orientation the cell may be communicating with. For a given orientation, the E1 model has twice as many iso-orientation zones per unit area as A1.

Ever since the discovery of orientation-specific cells in visual cortex there have been attempts to relate the distribution of cell selectivities to architectural features of the cortex. Hubel and Wiesel originally suggested that the orientation slabs followed the organization of the ocular dominance slabs[15]. The Braitenbergs suggested in their original model[7] that the centers might be identified with the giant cells of Meynert. Later centric models have identified the centers with the cytochrome oxidase regions, again relating the orientation map to the ocular dominance array, since the puffs themselves are closely related to this array.

While biologists have habitually related form to function, workers in machine vision have traditionally relied on general-purpose architectures to implement a variety of algorithms related to the processing of visual information[33]. More recently, many computer scientists designing artificial vision systems have turned their attention towards connectionist systems and neural networks. There is great interest in how the sensitivities to different features and how the selectivities to different values of those features may be embedded in the system architecture[34,35,36]. Linsker has proposed (this volume) that the development of feature spaces is a natural concomitance of layered networks, providing a generic organizing principle for networks. Our work deals with more specific cortical architectonics, but we are convinced that the study of the cortical layout of feature maps will provide important insights for the design of artificial systems.

## References

1. D. Hubel & T. Wiesel, *J. Physiol.* (*Lond.*) **160,** 106 (1962).
2. D. Hubel & T. Wiesel, *J. Physiol.* (*Lond.*) **195,** 225 (1968).
3. D. Hubel & T. Wiesel, *Proc. Roy. Soc. Lond. B* **198,** 1 (1977).
4. N. Swindale, *Proc. Roy. Soc. Lond. B* **215,** 211 (1982).
5. R.Linsker, *Proc. Natl. Acad. Sci. USA* **83,** 8779 (1986).
6. R. Soodak, *Proc. Natl. Acad. Sci. USA* **84,** 3936 (1987).

7. V. Braitenberg & C. Braitenberg, *Biol. Cyber.* **33**, 179 (1979).
8. B. Dow & R. Bauer, *Biol. Cyber.* **49**, 189 (1984).
9. K. Gotz, *Biol. Cyber.* **56**, 107 (1987).
10. P. Daniel & D. Whitteridge, *J. Physiol. (Lond.)* **159**, 302 (1961).
11. B. Dow, R. Vautin & R. Bauer, *J. Neurosci.* **5**, 890 (1985).
12. R.B. Tootell, M.S. Silverman, E. Switkes & R. DeValois, *Science* **218**, 902 (1982).
13. D.C. Van Essen, W.T. Newsome & J.H. Maunsell, *Vision Research* **24**, 429 (1984).
14. D. Hubel & T. Wiesel, *J. Comp. Neurol.* **158**, 295 (1974).
15. D. Hubel & T. Wiesel, *J. Comp. Neurol.* **146**, 421 (1972).
16. D. Hubel, *Nature* **299**, 515 (1982).
17. M. Livingstone & D. Hubel, *J. Neurosci.* **4**, 309 (1984).
18. R. Bauer, B. Dow, A. Snyder & R. Vautin, *Exp. Brain Res.* **50**, 133 (1983).
19. D. Hubel & T. Wiesel, *J. Comp. Neurol.* **158**, 267 (1974).
20. N. Swindale, in *Models of the Visual Cortex*, D. Rose & V. Dobson, eds., (Wiley, 1985), p. 452.
21. S. LeVay, D. Hubel, & T. Wiesel, *J. Comp. Neurol.* **159**, 559 (1975).
22. D. Hubel, T. Wiesel & M. Stryker, *J. Comp. Neurol.* **177**, 361 (1978).
23. T. Elsdale & F. Wasoff, *Wilhelm Roux's Archives* **180**, 121 (1976).
24. M.T. Wong-Riley, *Brain Res.* **162**, 201 (1979).
25. A. Humphrey & A. Hendrickson, *J. Neurosci.* **3**, 345 (1983).
26. E. Carroll & M. Wong-Riley, *J. Comp. Neurol.* **222**, 1 (1984).
27. J. Horton, *Proc. Roy. Soc. Lond. B* **304**, 199 (1984).
28. V. Braitenberg, in *Models of the Visual Cortex*, p. 479.
29. G. Mitchison, in *Models of the Visual Cortex*, p. 443.
30. C. Michael, *Vision Research* **25** 415 (1985).
31. S. Kirkpatrick, M. Gelatt & M. Vecchi, *Science* **220**, 671 (1983).
32. S. Tieman & H. Hirsch, in *Models of the Visual Cortex*, p. 432.
33. D. Ballard & C. Brown *Computer Vision* (Prentice-Hall, N.J., 1982).
34. D. Ballard, G. Hinton, & T. Sejnowski, *Nature* **306**, 21 (1983).
35. D. Ballard, *Behav. and Brain Sci.* **9**, 67 (1986).
36. D. Walters, *Proc. First Int. Conf. on Neural Networks* (June 1987).
